# Time-Warping Network :
# A Hybrid Framework for Speech Recognition

**Esther Levin**        **Roberto Pieraccini**        **Enrico Bocchieri**

AT&T Bell Laboratories
Speech Research Department
Murray Hill, NJ 07974 USA

## ABSTRACT

Recently, much interest has been generated regarding speech recognition systems based on Hidden Markov Models (HMMs) and neural network (NN) hybrids. Such systems attempt to combine the best features of both models: the temporal structure of HMMs and the discriminative power of neural networks. In this work we define a time-warping (TW) neuron that extends the operation of the formal neuron of a back-propagation network by warping the input pattern to match it optimally to its weights. We show that a single-layer network of TW neurons is equivalent to a Gaussian density HMM-based recognition system, and we propose to improve the discriminative power of this system by using back-propagation discriminative training, and/or by generalizing the structure of the recognizer to a multi-layered net. The performance of the proposed network was evaluated on a highly confusable, isolated word, multi speaker recognition task. The results indicate that not only does the recognition performance improve, but the separation between classes is enhanced also, allowing us to set up a rejection criterion to improve the confidence of the system.

## I. INTRODUCTION

Since their first application in speech recognition systems in the late seventies, hidden Markov models have been established as a most useful tool, mainly due to their ability to handle the sequential dynamical nature of the speech signal. With the revival of connectionism in the mid-eighties, considerable interest arose in applying artificial neural networks for speech recognition. This interest was based on the discriminative power of NNs and their ability to deal with non-explicit knowledge. These two paradigms, namely HMM and NN, inspired by different philosophies, were seen at first as different and competing tools. Recently, links have been established between these two paradigms, aiming at a hybrid framework in which the advantages of the two models can be combined. For example, Bourlard and Wellekens [1] showed that neural

networks with proper architecture can be regarded as non-parametric models for computing "discriminant probabilities" related to HMM. Bridle [2] introduced "Alpha-nets", a recurrent neural architecture that implements the alpha computation of HMM, and found connections between back-propagation [3] training and discriminative HMM parameter estimation. Predictive neural nets were shown to have a statistical interpretation [4], generalizing the conventional hidden Markov model by assuming that the speech signal is generated by nonlinear dynamics contaminated by noise.

In this work we establish one more link between the two paradigms by introducing the time-warping network (TWN) that is a generalization of both an HMM-based recognizer and a back-propagation net. The basic element of such a network, a *time-warping neuron*, generalizes the function of a formal neuron by warping the input signal in order maximize its activation. In the special case of network parameter values, a single-layered network of time-warping (TW) neurons is equivalent to a recognizer based on Gaussian HMMs. This equivalence of the HMM-based recognizer and single-layer TWN suggests ways of using discriminative neural tools to enhance the performance of the recognizer. For instance, a training algorithm, like back-propagation, that minimizes a quantity related to the recognition performance, can be used to train the recognizer instead of the standard non-discriminative maximum likelihood training. Then, the architecture of the recognizer can be expanded to contain more than one layer of units, enabling the network to form discriminant feature detectors in the hidden layers.

This paper is organized as follows: in the first part of Section 2 we describe a simple HMM-based recognizer. Then we define the time-warping neuron and show that a single-layer network built with such neurons is equivalent to the HMM recognizer. In Section 3 two methods are proposed to improve the discriminative power of the recognizer, namely, adopting neural training algorithms and extending the structure of the recognizer to a multi-layer net. For special cases of such multi-layer architecture such net can implement a conventional or weighted [5] HMM recognizer. Results of experiments using a TW network for recognition of the English E-set are presented in Section 4. The results indicate that not only does the recognition performance improve, but the separation between classes is enhanced also, allowing us to set up a rejection criterion to improve the confidence of the system. A summary and discussion of this work are included in Section 5.

## II.  THE MODEL

In this section first we describe the basic HMM-based speech recognition system that is used in many applications, including isolated and connected word recognition [6] and large vocabulary subword-based recognition [7]. Though in this paper we treat the case of isolated word recognition, generalization to connected speech can be made like in [6,7]. In the second part of this section we define a single-layered time-warping network and show that it is equivalent to the HMM based recognizer when certain conditions constraining the network parameter values apply.

### II.1  THE HIDDEN MARKOV MODEL-BASED RECOGNITION SYSTEM

A HMM-based recognition system consists of $K$ $N$-state HMMs, where $K$ is the vocabulary size (number of words or subword units in the defined task). The $k$-th HMM, $\Omega^k$, is associated with the $k$-th word in the vocabulary and is characterized by a matrix $A^k = \{a_{ij}^k\}$ of transition probabilities between states,

$$a_{ij}^k = Pr(s_t = j \mid s_{t-1} = i) , \ 0 \leq i \leq N , \ 1 \leq j \leq N , \qquad (1)$$

where $s_t$ denotes the active state at time $t$ ($s_0 = 0$ is a dummy initial state) and by a set of emission probabilities (one per state):

$$Pr(X_t \mid s_t=i) = \frac{1}{\sqrt{2\pi} \parallel \Sigma_i^k \parallel^2} \exp\left[-\frac{1}{2}(X_t-\mu_i^k)^* (\Sigma_i^k)^{-1} (X_t-\mu_i^k)\right], \quad i=1, \cdots, N, \quad (2)$$

where $X_t$ is the $d$-dimensional observation vector describing some parametric representation of the $t$-th frame of the spoken token, and $()^*$ denotes the transpose operation.

For the case discussed here, we concentrate on *strictly left-to-right* HMMs, where $a_{ij}^k \neq 0$ only if $j=i$ or $j=i+1$, and a simplified case of (2) where all $\Sigma_i^k = I_d$, the $d$=dimensional unit matrix.

The system recognizes a speech token of duration $T$, $X=\{X_1, X_2, \cdots, X_T\}$, by classifying the token into the class $k_0$ with the highest likelihood $L^{k_0}(X)$,

$$k_0 = arg \max_{1 \leq k \leq K} L^k(X). \quad (3)$$

The likelihood $L^k(X)$ is computed for the $k$-th HMM as

$$L^k(X) = \max_{\{i_1, \cdots, i_T\}} \log[Pr(X \mid \Omega^k, s_i=i_1, \cdots, s_T=i_t)] \quad (4)$$

$$= \max_{\{i_1, \cdots, i_T\}} \sum_{t=1}^{T} \frac{-1}{2} \parallel X_t-\mu_{i_t}^k \parallel^2 + \log a_{i_{t-1}i_t}^k - \log 2\pi.$$

The state sequence that maximizes (4) is found by using the Viterbi [8] algorithm.

## II.2   THE EQUIVALENT SINGLE-LAYER TIME-WARPING NETWORK

A single-layer TW network is composed of $K$ TW neurons, one for each word in the vocabulary. The TW neuron is an extension of a formal neuron that can handle dynamic and temporally distorted patterns. The $k$-th TW neuron, associated with the $k$-th vocabulary word, is characterized by a bias $w_0^k$ and a set of weights, $\hat{W}^k = \{W_1, W_2, \cdots, W_N\}$, where $W_i$ is a column vector of dimensionality $d+2$. Given an input speech token of duration $T$, $X=\{X_1, X_2, \cdots, X_T\}$, the output activation $y^k$ of the $k$-th unit is computed as

$$y^k = g\left(\sum_{t=1}^{T} \hat{X}_t^* \cdot \hat{W}_{i_t}^k + w_0^k\right) = g\left(\sum_{j=1}^{N} \left(\sum_{t:i_t=j} \hat{X}_t^*\right) \cdot \hat{W}_j^k + w_0^k\right), \quad (5)$$

where $g(\cdot)$ is a sigmoidal, smooth, strictly increasing nonlinearity, and $\hat{X}_t^* = [X_t^*, 1, 1]$ is an $d+2$ - dimensional augmented input vector. The corresponding indices $i_t$, $t=1, \cdots, T$ are determined by the following condition:

$$\{i_1, \cdots, i_T\} = argmax \sum_{t=1}^{T} \hat{X}_t^* \cdot \hat{W}_{i_t}^k + w_0^k. \quad (6)$$

In other words, a TW neuron warps the input pattern to match it optimally to its weights (6) and computes its output using this warped version of the input (5). The time-warping process of (6) is a distinguishing feature of this neural model, enabling it to deal with the dynamic nature of a speech signal and to handle temporal distortions.

All TW neurons in this single-layer net recognizer receive the same input speech token $X$. Recognition is performed by selecting the word class corresponding to the neuron with the maximal output activation.

It is easy to show that when

$$[\hat{W}_j^k]^* = [[\mu_j^k]^*, -\frac{1}{2} \parallel \mu_j^k \parallel^2, \log a_{j,j}^k], \quad (7a)$$

and

$$w_0^k = \sum_{j=1}^{N} \log a_{j,j-1}^k - \log a_{j,j}^k \qquad (7b)$$

this network is equivalent to an HMM-based recognition system, with $K$ $N$-state HMMs, as described above.[1]

This equivalent neural representation of an HMM-based system suggests ways of improving the discriminative power of the recognizer, while preserving the temporal structure of the HMM, thus allowing generalization to more complicated tasks (e.g., continuous speech, subword units, etc.).

## III. IMPROVING DISCRIMINATION

There are two important differences between the HMM-based system and a neural net approach to speech recognition that contribute to the improved discrimination power of the latter, namely, training and structure.

### III.1 DISCRIMINATIVE TRAINING

The HMM parameters are usually estimated by applying the maximum likelihood approach, using only the examples of the word represented by the model and disregarding the rival classes completely. This is a non-discriminative approach: the learning criterion is not directly connected to the improvement of recognition accuracy. Here we propose to enhance the discriminative power of the system by adopting a neural training approach.

NN training algorithms are based on minimizing an error function $E$, which is related to the performance of the network on the training set of labeled examples, $\{X^l, Z^l\}$, $l=1, \cdots, L$, where $Z^l = [z_1^l, \cdots, z_K^l]^*$ denotes the vector of target neural outputs for the $l$-th input token. $Z^l$ has $+1$ only in the entry corresponding to the right word class, and $-1$ elsewhere. Then,

$$E = \sum_{l=1}^{L} E^l(Z^l, Y^l), \qquad (8)$$

where $Y^l = [y_1^l, \cdots, y_K^l]^*$ is a vector of neural output activations for the $l$-th input token, and $E^l(Z^l, Y^l)$ measures the distortion between the two vectors. One choice for $E^l(Z^l, Y^l)$ is a quadratic error measure, i.e., $E^l(Z^l, Y^l) = \| Z^l - Y^l \|^2$. Other choices include the cross-entropy error [9] and the recently proposed discriminative error functions, which measure the misclassification rate more directly [10].

The gradient based training algorithms (such as back-propagation) modify the parameters of the network after presentation of each training token to minimize the error (8). The change in the $j$-th weight subvector of the $k$-th model after presentation of the $l$-th training token, $\Delta^l W_j^k$ is inversely proportional to the derivative of the error $E^l$ with respect to this weight subvector,

$$\Delta^l W_j^k = -\alpha \frac{\partial E^l}{\partial W_j^k} = -\alpha \sum_{m=1}^{K} \frac{\partial E^l}{\partial y_m^l} \frac{\partial y_m^l}{\partial W_j^k}, \quad 1 \le j \le N, \ 1 \le k \le K, \qquad (9)$$

where $\alpha > 0$ is a step-size, resulting in an updated weight vector $[\hat{W}_j^k]^* = [[W_j^k + \Delta W_j^k]^*, -\frac{1}{2} \| W_j^k + \Delta W_j^k \|^2, \log a_{j,j}^k]$. To compute the terms $\frac{\partial y_m^l}{\partial W_j^k}$

---

1.  With minor changes we can show equivalence to a general Gaussian HMM, where the covariance matrices are not restricted to be the unit matrix.

we have to consider (5) and (6) that define the operation of the neuron. Equation (6) expresses the dependence of the warping indices $i_1, \cdots, i_T$ on $W_j^k$. In the proposed learning rule we compute the gradient for the quadratic error criterion using only (5).

$$\Delta^l W_j^k = \alpha (z_k - y_k) g'(\cdot) \sum_{t:i_t=j} X_t^l - W_j^k , \qquad (10)$$

where the values of $i_t$ fulfill condition (6). Although the weights do not change according to the exact gradient descent rule (since (6) is not taken into account for back-propagation) we found experimentally that the error made by the network always decreases after the weight update. This fact also can be proved when certain conditions restricting the step-size $\alpha$ hold, and we conjecture that it is always true for $\alpha > 0$.

## III.2 THE STRUCTURE OF THE RECOGNIZER

When the equivalent neural representation of the HMM-based recognizer is used, there exists a natural way of adaptively increasing the complexity of the decision boundaries and developing discriminative feature detectors. This can be done by extending the structure of the recognizer to a multi-layered net. There are many possible architectures that result from such an extension by changing the number of hidden layers, as well as the number and the type (i.e., standard or TW ) of neurons in the hidden layers. Moreover, the role of the TW neurons in the first hidden layer is different now: they are no longer class representatives, as in a single-layered net, but just abstract computing elements with built-in time scale normalization. In this work we investigate only a simple special case of such multi-layered architecture. The multi-layered network we use has a single hidden layer, with $N \times K$ TW neurons. Each hidden neuron corresponds to one state of one of the original HMMs, and is characterized by a weight vector $W_j^k$ and a bias $w_j^k$. The output activation $h_j^k$ of the neuron is given as

$$h_j^k = g(u_j^k), \qquad (11)$$

where

$$u_j^k = \sum_{t:i_t=j} \hat{X}_t^* \hat{W}_j^k + w_j^k , $$

and

$$\{ i_1, \cdots, i_T \} = argmax \sum_{j=1}^{N} u_j^k .$$

The output layer is composed of $K$ standard neurons. The activation of output neurons $y^k, k=1, \ldots, K$, is determined by the hidden layer neurons activations as

$$y^k = g(H^* V^k + v_k), \qquad (12)$$

where $V^k$ is a $N \times K$ dimensional weight vector, $H$ is the vector of hidden neurons activation, and $v_k$ is a bias term.

In a special case of parameter values, when $\hat{W}_j^k$ satisfy the conditions (7a,b) and

$$w_j^k = \log a_{j,j-1}^k - \log a_{j,j}^k , \qquad (13)$$

the activation $h_j^k$ corresponds to an accumulated $j$-th state likelihood of the $k$-th HMM and the network implements a weighted [5] HMM recognizer where the connection weight vectors $V^k$ determine the relative weights assigned to each state likelihood in the final classification. Such network can learn to adopt these weights to enhance discrimination by giving large positive weights to states that contain information important for discrimination and ignoring (by forming zero or close to zero weights) those states that do not contribute to discrimination. A back-propagation algorithm

can be used for training this net.

## IV. EXPERIMENTAL RESULTS

To evaluate the effectiveness of the proposed TWN, we conducted several experiments that involved recognition of the highly confusable English E-set (i.e., /b, c, d, e, g, p, t, v, z/). The utterances were collected from 100 speakers, 50 males and 50 females, each speaking every word in the E-set twice, once for training and once for testing. The signal was sampled at 6.67 kHz. We used 12 cepstral and 12 delta-cepstral LPC-derived [11] coefficients to represent each 45 msec frame of the sampled signal.

We used a baseline conventional HMM-based recognizer to initialize the TW network, and to get a benchmark performance. Each strictly left-to-right HMM in this system has five states, and the observation densities are modeled by four Gaussian mixture components. The recognition rates of this system are 61.7% on the test data, and 80.2% on the training data.

**Experiment with single-layer TWN:** In this experiment the single-layer TW network was initialized according to (7), using the parameters of the baseline HMMs. The four mixture components of each state were treated as a fully connected set of four states, with transition probabilities that reflect the original transition probabilities and the relative weights of the mixtures. This corresponds to the case in which the local likelihood is computed using the dominant mixture component only. The network was trained using the suggested training algorithm (10), with quadratic error function. The recognition rate of the trained network increased to 69.4% on the test set and 93.6% on the training set.

**Experiment with multi-layer TWN:** In this experiment we used the multi-layer network architecture described in the previous section. The recognition performance of this network after training was 74.4% on the test set and 91% on the training set.

Figures 1, 2, and 3 show the recognition performance of a single-layer TWN, initialized by a baseline HMM, the trained single-layer TWN, and the trained multi-layer TWN, respectively. In these figures the activation of the unit representing the correct class is plotted against the activation of the *best wrong* unit (i.e., the incorrect class with the highest score) for each input utterance. Therefore, the utterances that correspond to the marks above the diagonal line are correctly recognized, and those under it are misclassified. The most interesting observation that can be made from these plots is the striking difference between the multi-layer and the single-layer TWNs. The single-layer TWNs in Figures 1 and 2 (the baseline and the trained) exhibit the same typical behavior when the utterances are concentrated around the diagonal line. For the multi-layer net, the utterances that were recognized correctly tend to concentrate in the upper part of the graph, having the correct unit activation close to 1.0. This property of a multi-layer net can be used for introducing error rejection criterions: utterances for which the difference between the highest activation and second high activation is less than a prescribed threshold are rejected. In Figure 4 we compare the test performance of the multi-layer net and the baseline system, both with such rejection mechanism, for different values of rejection threshold. As expected, the multi-layer net outperforms the baseline recognizer, by showing much smaller misclassification rate for the same number of rejections.

## V. SUMMARY AND DISCUSSION

In this paper we established a hybrid framework for speech recognition, combining the characteristics of hidden Markov models and neural networks. We showed that a HMM-based recognizer has an equivalent representation as a single-layer network composed of time-warping neurons, and proposed to improve the discriminative power of the recognizer by using back-propagation training and by generalizing the structure of the recognizer to a multi-layer net. Several experiments were conducted for testing

the performance of the proposed network on a highly confusable vocabulary (the English E-set). The recognition performance on the test set of a single-layer TW net improved from 61% (when initialized with a baseline HMMs) to 69% after training. Expending the structure of the recognizer by one more layer of neurons, we obtained further improvement of recognition accuracy up to 74.4%. Scatter plots of the results indicate that in the multi-layer case, there is a qualitative change in the performance of the recognizer, allowing us to set up a rejection criterion to improve the confidence of the system.

## References

1. H. Bourlard, C.J. Wellekens, "Links between Markov models and multilayer perceptrons," *Advances in Neural Information Processing Systems,* pp.502-510, Morgan Kauffman, 1989.
2. J.S. Bridle, "Alphanets: a recurrent 'neural' network architecture with a hidden Markov model interpretation," *Speech*Communication, April 1990.
3. D.E. Rumelhart, G.E. Hinton and R.J. Williams, "Learning internal representation by error propagation," *Parallel Distributed Processing: Exploration in the Microstructure of Cognition,* MIT Press, 1986.
4. E. Levin, "Word recognition using hidden control neural architecture," *Proc. of ICASSP,* Albuquerque, April 1990.
5. K.-Y. Su, C.-H. Lee, "Speech Recognition Using Weighted HMM and Subspace Projection Approaches," *Proc of ICASSP,* Toronto, 1991.
6. L. R. Rabiner, "A tutorial on hidden Markov models and selected applications in speech recognition," *Proc. of IEEE,* vol. 77, No. 2, pp. 257-286, February 1989.
7. C.-H. Lee, L. R. Rabiner, R. Pieraccini, J. G. Wilpon, "Acoustic Modeling for Large Vocabulary Speech Recognition," *Computer Speech and Language,* 1990, No. 4, pp. 127-165.
8. G.D. Forney, "The Viterbi algorithm," *Proc. IEEE,* vol. 61, pp. 268-278, Mar. 1973.
9. S.A. Solla, E. Levin, M. Fleisher, "Improved targets for multilayer perceptron learning," *Neural Networks Journal,* 1988.
10. B.-H. Juang, S. Katagiri, "Discriminative Learning for Minimum Error Classification," *IEEE Trans. on SP,* to be published.
11. B.S. Atal, "Effectiveness of linear prediction characteristics of the speech wave for automatic speaker identification and verification," *J. Acoust. Soc. Am.,* vol. 55, No. 6, pp. 1304-1312, June 1974.

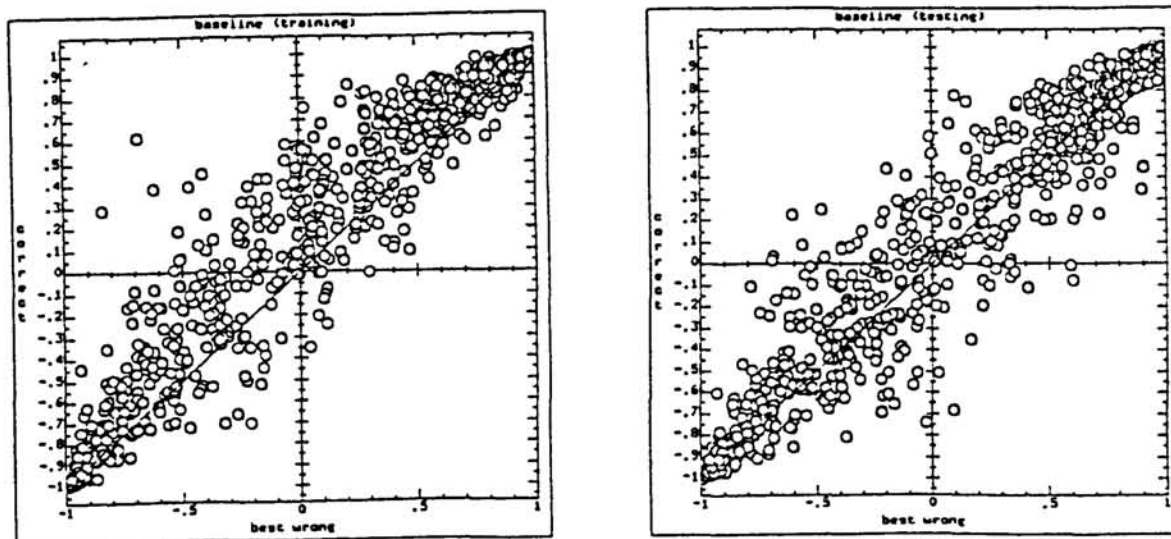

Figure 1: Scatter plot for baseline recognizer

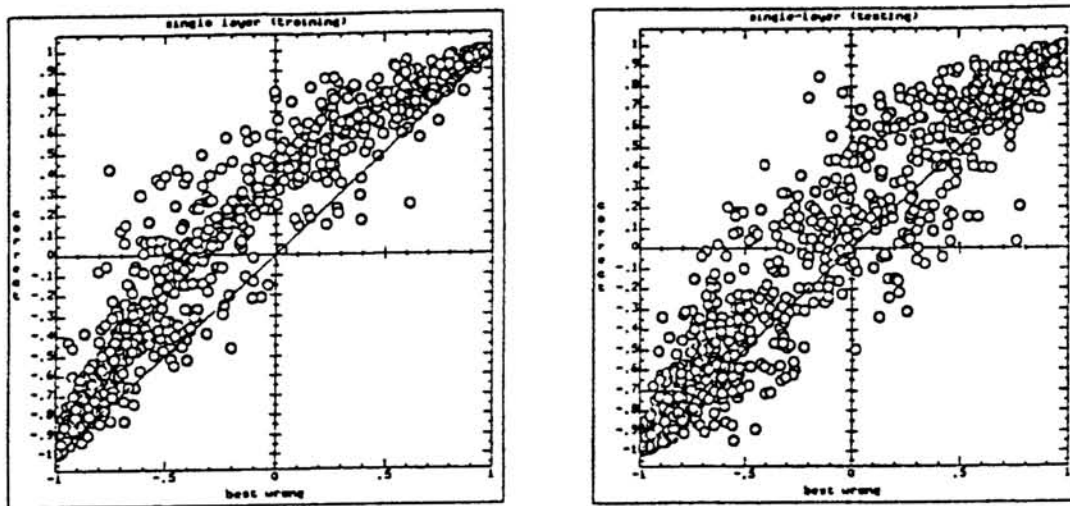

Figure 2: Scatter plot for trained single-layer TWN

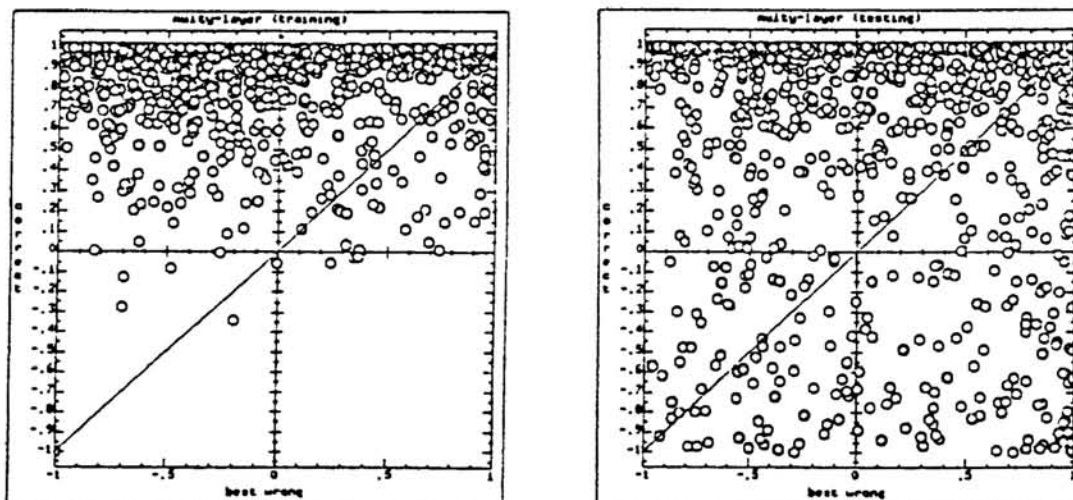

Figure 3: Scatter plot for multi-layer TWN

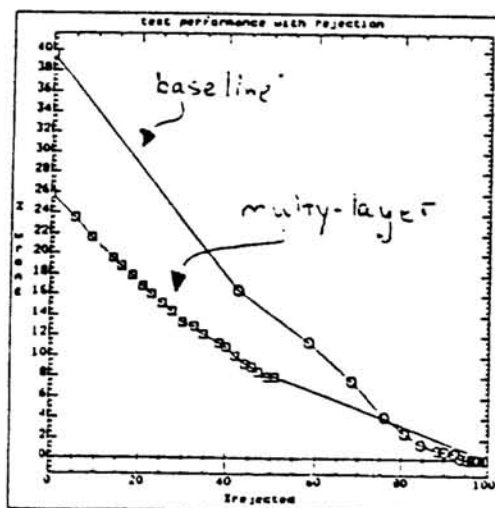

Figure 4: Rejection performance of baseline recognizer and the multi-layer TWN